# Classification of Multi-Spectral Pixels by the Binary Diamond Neural Network

**Yehuda Salu**

Department of Physics and CSTEA, Howard University, Washington, DC 20059

## Abstract

A new neural network, the Binary Diamond, is presented and its use as a classifier is demonstrated and evaluated. The network is of the feed-forward type. It learns from examples in the 'one shot' mode, and recruits new neurons as needed. It was tested on the problem of pixel classification, and performed well. Possible applications of the network in associative memories are outlined.

## 1    INTRODUCTION: CLASSIFICATION BY CLUES

Classification is a process by which an item is assigned to a class. Classification is widely used in the animal kingdom. Identifying an item as food is classification. Assigning words to objects, actions, feelings, and situations is classification. The purpose of this work is to introduce a new neural network, the Binary Diamond, which can be used as a general purpose classification tool. The design and operational mode of the Binary Diamond are influenced by observations of the underlying mechanisms that take place in human classification processes.

An item to be classified consists of **basic features**. Any arbitrary combination of basic features will be called a **clue**. Generally, an item will consist of many clues. Clues are related not only to the items which contain them, but also to the classes. Each class, that resides in the memory, has a list of clues which are associated with it. These clues

are the basic building blocks of the classification rules. **A classification rule** for a class X would have the following general form:

**Classification rule** : If an item contains clue $X_1$, or clue $X_2$,..., or clue $X_n$, and if it does not contain clue $X_1$, nor clue $X_2$, ..., nor clue $X_m$, it is classified as belonging to class X.

Clues $X_1$,...,$X_n$ are the **excitatory clues** of class X, and clues $X_1$,...,$X_m$ are the **inhibitory clues** of class X.

When classifying an item, we first identify the clues that it contains. We then match these clues with the classification rules, and find the class of the item. It may happen that a certain item satisfies classification rules of different classes. Some of the clues match one class, while others match another. In such cases, a second set of rules, **disambiguation rules**, are employed. These rules select one class out of those tagged by the classification rules. The disambiguation rules rely on a hierarchy that exists among the clues, a hierarchy that may vary from one classification scheme to another. For example, in a certain hierarchy clue A is considered more reliable than clue B, if it contains more features. In a different hierarchy scheme, the most frequent clue is considered the most reliable. In the disambiguation process, the most reliable clue, out of those that has actively contributed to the classification, is identified and serves as the pointer to the selected class. This classification approach will be called **classification by clues (CBC)**.

The classification rules may be 'loaded' into our memory in two ways. First, the precise rules may be spelled out and recorded (e.g. 'A red light means stop'). Second, we may learn the classification rules from examples presented to us, utilizing innate common sense learning mechanism. These mechanisms enable us to deduce from the examples presented to us, what clues should serve in the classification rules of the adequate classes, and what clues have no specificity, and should be ignored. For example, by pointing to a red balloon and saying red, an infant may associate each of the stimuli red and balloon as pointers to the word red. After presenting a red car, and saying red, and presenting a green balloon and saying green, the infant has enough information to deduce that the stimulus red is associated with the word red, and the stimulus balloon should not be classified as red.

## 2    THE BINARY DIAMOND

### 2.1    STRUCTURE

In order to perform a **CBC** in a systematic way, all the clues that are present in the item to be classified have to be identified first, and then compared against the classification rules. The Binary Diamond enables carrying these tasks fast and

effectively. Assume that there are N different basic features in the environment. Each feature can be assigned to a certain bit in an N dimensional binary vector. An item will be represented by turning-on (from the default value of 0 to the value of 1) all the bits that correspond to basic features, that are present in the item. The total number of possible clues in this environment is at most $2^N$. One way to represent these possible clues is by a lattice, in which each possible clue is represented by one node. The Binary Diamond is a lattice whose nodes represent clues. It is arranged in layers. The first (bottom) layer has N nodes that represent the basic features in the environment. The second layer has $N\cdot(N-1)/2$ nodes that represent clues consisting of 2 basic features. The K'th layer has nodes that represent clues, which consist of K basic features. Nodes from neighboring layers which represent clues that differ by exactly one basic feature are connected by a line. Figure 1 is a diagram of the Binary Diamond for N=4.

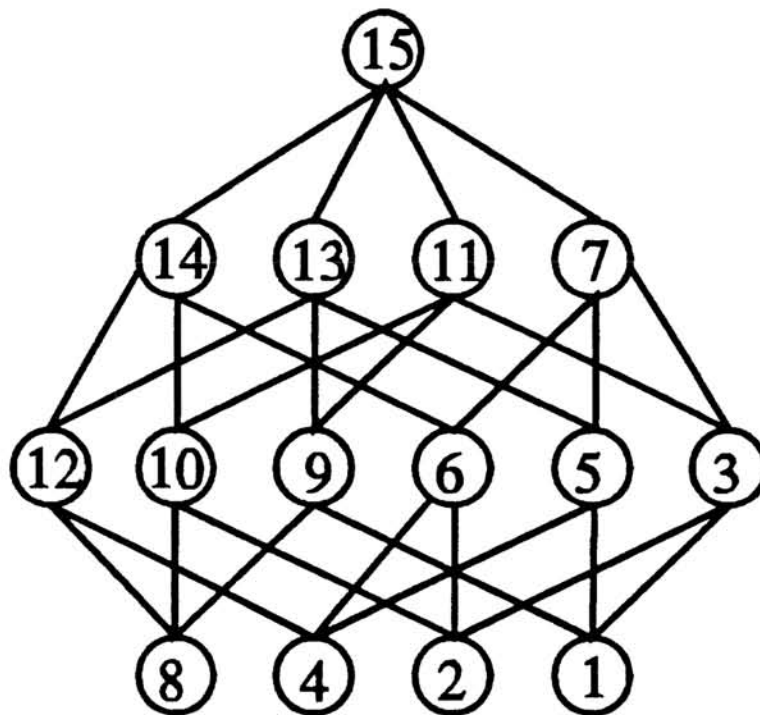

Figure 1: The Binary Diamond of order 4. The numbers inside the nodes are the binary codes for the feature combination that the node represents, e.g 1 < = > (0,0,0,1), 5< = > (0,1,0,1), 14 < = > (1,1,1,0), 15 < = > (1,1,1,1).

## 2.2   THE BINARY DIAMOND NEURAL NETWORK

The Binary Diamond can be turned into a feed-forward neural network by treating each node as a neuron, and each line as a synapse leading from a neuron in a lower layer (k) to a neuron in the higher layer (k+1). All synaptic weights are set to 0.6, and

all thresholds are set to 1, in a standard Pitts McCulloch neuron. The output of a firing neuron is 1. An item is entered into the network by turning-on the neurons in the first layer, that represent the basic features constituting this item. Signals propagate forward one layer at a time tick, and neurons stay active for one time tick. It is easy to verify that all the clues that are part of the input item, and only such clues, will be turned on as the signals propagate in the network. In other words, the network identifies all the clues in the item to be classified. An item consisting of M basic features will activate neurons in the first M layers. The activated neuron in the M'th layer is the representation of the entire item. As an example, consider the input item with feature vector (0,1,1,1), using the notations of figure 1. It is entered by activating neurons 1, 2, and 4 in the first layer. The signals will propagate to neurons 3, 5, 6, and 7, which represent all the clues that the input item contains.

## 2.3    INCORPORATING CLASS INFORMATION

Each neuron in the Binary Diamond represent a possible clue in the environment spun by N basic features. When an item is entered in the first layer, all the clues that it contains activate their representing neurons in the upper layers. This is the first step in the classification process. Next, these clues have to point to the appropriate class, based upon the classification rule. The possible classes are represented by neurons outside of the Binary Diamond. Let x denote the neuron, outside the Binary Diamond, that represents class X. An excitatory clue $X_i$ (from the Binary Diamond) will synapse onto x with a synaptic weight of 1. An inhibitory clue $X_j$ (in the Binary Diamond) will synapse onto x with an inhibitory weight of -z, where z is a very large number (larger than the maximum number of clues that may point to a class). This arrangement ensures that the classification rule formulated above is carried out. In cases of ambiguity, where a number of classes have been activated in the process, the class that was activated by the clue in the highest layer will prevail. This clue has the largest number of features, as compared with the other clues that actively participated in the classification.

## 2.4    GROWING A BINARY DIAMOND

A possible limitation on the processes described in the two previous sections is that, if there are many basic features in the environment, the $2^N$ nodes of the Binary Diamond may be too much to handle. However, in practical situations, not all the clues really occur, and there is no need to actually represent all of them by nodes. One way of taking advantage of this simplifying situation is to grow the network one **event** (a training item and its classification) at a time. At the beginning, there is just the first layer with N neurons, that represent the N basic features. Each event adds its neurons to the network, in the exact positions that they would occupy in the regular

Binary Diamond. A clue that has already been represented in previous events, is not duplicated. After the new clues of the event have been added to the network, the information about the relationships between clues and classes is updated. This is done for all the clues that are contained in the new event. The new neurons send synapses to the neuron that represent the class of the current event. Neurons of the current event, that took part in previous events, are checked for consistency. If they point to other classes, their synapses are cut-off. They have just lost their specificity. It should be noted that there is no need to present an event more than one time for it to be correctly recorded ('one shot learning'). A new event will never adversely interfere with previously recorded information. Neither the order of presenting the events, nor repetitions in presenting them will affect the final structure of the network. Figure 2 illustrates how a Binary Diamond is grown. It encodes the information contained in two events, each having three basic features, in an environment that has four basic features. The first event belongs to class A, and the second to class B.

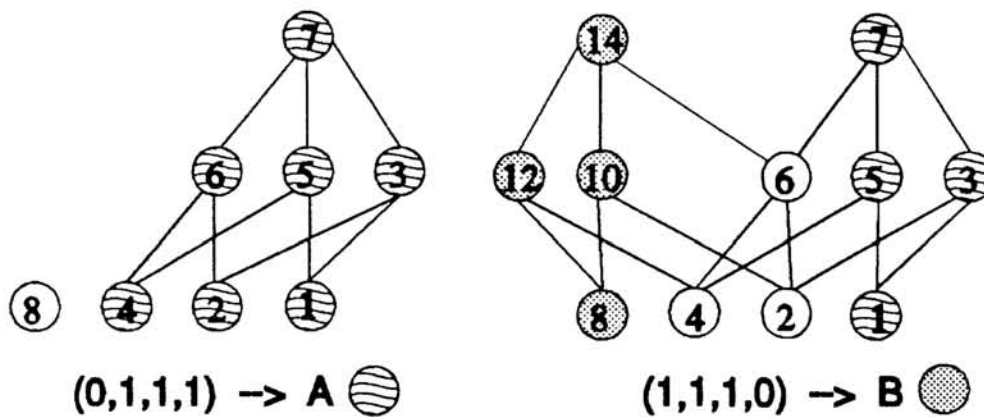

Figure 2. Growing a Binary Diamond. Left: All the feature combinations of the three-feature item (0,1,1,1) are represented by a 3'rd order Binary Diamond, which is grown from the basic features represented by neurons 1, 2, and 4. All these combinations, marked by a wavy background, are, for the time being, specific clues to class A. Right: The three-feature item, (1,1,1,0) is added, as another 3'rd order Binary Diamond. At this point, only neurons 1,3,5,and 7 represent specific clues to class A. Neurons 8,10,12, and 14 represent specific clues to class B, and neurons 2,4, and 6 represent non-specific clues.

# 3    CLASSIFICATION OF MULTI-SPECTRAL PIXELS

## 3.1    THE PROBLEM

Spectral information of land pixels, which is collected by satellites, is used in preparation of land cover maps and similar applications. Depending on the satellite and its instrumentation, the spectral information consists of the intensities of several

light bands, usually in the visible and infra-red ranges, which have been reflected from the land pixels. One method of classification of such pixels relies on independent knowledge of the land cover of some pixels in the scene. These classified pixels serve as the training set for a classification algorithm. Once the algorithm is trained, it classifies the rest of the pixels.

The actual problem described here involves testing the Binary Diamond in a pixel classification problem. The tests were done on four scenes from the vicinity of Washington DC, each consisting of approximately 22,000 pixels. The spectral information of each pixel consisted of intensities of four spectral bands, as collected by the Thematic Mapper of the Landsat 4 satellite. Ground covers of these scenes were determined independently by ground and aerial surveys. There were 17 classes of ground covers. The following list gives the number of pixels per class in one of the scenes. The distributions in the other scenes were similar.

1) water (28). 2) miscellaneous crops (299). 3) corn-standing (0). 4) corn-stubble (349). 5) shrub-land (515). 6) grass/ pasture (3,184). 7) soybeans (125). 8) bare-soil, clear land (535). 9) hardwood, canopy > 50% (10,169). 10) hardwood, canopy < 50% (945). 11) conifer forest (2,051). 12) mixed wood forest (616). 13) asphalt (390). 14) single family housing (2,220). 15) multiple family housing (26). 16) industrial/ commercial (118). 17) bare soil-plowed field (382). Total 21,952.

## 3.2   METHODS

Approximately 10% of the pixels in each of the four scenes were randomly selected to become a training set. Four Binary Diamond networks were grown, based on these four training sets. In the evaluation phase, each network classified each scene.

The intensity of the light in each band was discretized into 64 intervals. Each interval was considered as a basic feature. So, each pixel was characterized by four basic features (one for each band), out of 4x64=256 possible basic features. The first layer of the Binary Diamond consisted of 256 neurons, representing these basic features. Pixels of the training set were treated like events. They were presented sequentially, one at a time, for one time, and the neurons that represent their clues were added to the network, as explained in section 2.4. After the training phase, the rest of the pixels were presented, and the network classified them. The results of this classification were kept for comparisons with the observed ground cover values.

The same training sets were used to train two other classification algorithms; a backpropagation neural network, and a nearest neighbor classifier. The back-propagation network had four neurons in the input layer, each representing a spectral band. It had seventeen neurons in the output layer, each representing a class, and a hidden layer of ten neurons. The nearest neighbor classifier used the pixels of the training set as models. The Euclidean distance between the feature vector of a pixel to

be classified and each model pixel was computed. The pixel was classified according to the class of its closest model.

## 3.3    RESULTS

In **auto-classification**, the pixels of a scene are classified by an algorithm that was trained using pixels from the same scene. In **cross-classification**, the classification of a scene is done by an algorithm that was trained by pixels of another scene. It was found that in both auto-classification and cross-classification, the results depend on the consistency of the training set. Boundary pixels, which form the boundary (on the ground) between two classes, may contain a combination of two ground cover classes. If boundary pixels were excluded from the scene, the results of all the classification methods improved significantly. Table 1  compares the overall performance of the three classification methods in auto-classification and cross-classification, when only boundary pixels were considered. Similar ordering of the classification methods was obtained when all the pixels were considered.

|   | 1 | 2 | 3 | 4 |
|---|---|---|---|---|
| 1 | 83 | 58 | 71 | 74 |
| 2 | 41 | 78 | 50 | 44 |
| 3 | 49 | 48 | 75 | 52 |
| 4 | 54 | 44 | 57 | 76 |

Binary Diamond

|   | 1 | 2 | 3 | 4 |
|---|---|---|---|---|
| 1 | 83 | 41 | 46 | 61 |
| 2 | 27 | 73 | 28 | 17 |
| 3 | 43 | 38 | 62 | 35 |
| 4 | 52 | 36 | 39 | 70 |

Nearest Neighbor

|   | 1 | 2 | 3 | 4 |
|---|---|---|---|---|
| 1 | 73 | 60 | 33 | 64 |
| 2 | 25 | 55 | 38 | 26 |
| 3 | 33 | 38 | 52 | 37 |
| 4 | 48 | 43 | 42 | 60 |

Back-Propagation

Table 1:  The percent of correctly classified pixels for the implementations of the three methods, for non-boundary pixels only,  as tested on the four maps. Column's index is the training map, row's index is the testing map.

Table 2 compares the performances of the three methods class by class, as obtained in the classification of the first scene. Similar results were obtained for the other scenes.

| I= | 1 | 2 | 3 | 4 | 5 | 6 | 7 | 8 | 9 | 10 | 11 | 12 | 13 | 14 | 15 | 16 | 17 |
|---|---|---|---|---|---|---|---|---|---|---|---|---|---|---|---|---|---|
| BD | 48 | 10 | 0 | 33 | 7 | 44 | 10 | 53 | 88 | 37 | 34 | 5 | 32 | 69 | 33 | 34 | 41 |
| BDp | 57 | 8 | 0 | 48 | 10 | 14 | 10 | 58 | 87 | 37 | 35 | 6 | 30 | 69 | 42 | 25 | 27 |
| bNN | 63 | 54 | 0 | 72 | 47 | 19 | 52 | 77 | 60 | 63 | 48 | 60 | 70 | 43 | 64 | 62 | 72 |
| BP | 68 | 5 | 0 | 68 | 0 | 1 | 11 | 66 | 80 | 74 | 11 | 1 | 26 | 45 | 54 | 52 | 27 |

Table 2:  The percent of pixels from category I that have been classified as category I. Auto-classification of scene 1. All the pixels are included. BD = results of Binary Diamond where the feature vectors are in the standard Cartesian representation. BDp = results of Binary Diamond where the feature vectors are in four dimensional polar coordinates. bNN results of nearest neighbor, and BP of back-propagation.

The overall performance of the Binary Diamond was better than those of the nearest neighbor and the back-propagation classifiers. This was the case in auto-classification and in cross-classification, in scenes that included all the pixels, and in scenes that consisted only of non-boundary pixels. However, when comparing individual classes, it was found that different classes may have different best classifiers. In practical applications, the prices of correct or the wrong classifications of each class, as well as the frequency of the classes in the environment will determine the optimal classifier.

All the networks recruited their neurons as needed, during the training phase. They all started with 256 neurons in the first layer, and with seventeen neuron in the class layer, outside the Binary Diamond. At the end of the training phase of the first scene, The Binary Diamond consisted of 5,622 neurons, in four layers. This is a manageable number, and it is much smaller than the maximum number of possible clues, $64^4 = 2^{24}$.

## 4     OTHER APPLICATIONS OF THE BINARY DIAMOND

The Binary Diamond, as presented here, was the core of a network that was used as a classifier. Because of its special structure, the Binary Diamond can be used in other related problems, such as in associative memories. In associative memory, a presented clue has to retrieve all the basic features of an associated item. If we start from any node in the Binary Diamond, and cascade down in the existing lines, we reach all the basic features of this clue in the first layer. So, to retrieve an associated item, the signals of the input clue have first to climb up the binary diamond till they reach a node, which is the best generalization of this clue, and then to cascade down and to activate the basic features of this generalization. The synaptic weights in the upward direction can encode information about causality relationships and the frequency of co-activations of the pre and post-synaptic neurons. This information can be used in the retrieval of the most appropriate generalization to the given clue. An associative memory of this kind retrieves information in ways similar to human associative retrieval (paper submitted).

### REFERENCES

A reference list, as well as more details about pixel classification can be found in: Classification of Multi-Spectral Image Data by the Binary Diamond Neural Network and by Non-Parametric Pixel-by-Pixel Methods, by Yehuda Salu and James Tilton. IEEE Transactions On Geoscience And Remote Sensing, 1993 (in press).